# Replicator Equations, Maximal Cliques, and Graph Isomorphism

**Marcello Pelillo**
Dipartimento di Informatica
Università Ca' Foscari di Venezia
Via Torino 155, 30172 Venezia Mestre, Italy
E-mail: `pelillo@dsi.unive.it`

## Abstract

We present a new energy-minimization framework for the graph isomorphism problem which is based on an equivalent maximum clique formulation. The approach is centered around a fundamental result proved by Motzkin and Straus in the mid-1960s, and recently expanded in various ways, which allows us to formulate the maximum clique problem in terms of a standard quadratic program. To solve the program we use "replicator" equations, a class of simple continuous- and discrete-time dynamical systems developed in various branches of theoretical biology. We show how, despite their inability to escape from local solutions, they nevertheless provide experimental results which are competitive with those obtained using more elaborate mean-field annealing heuristics.

## 1  INTRODUCTION

The graph isomorphism problem is one of those few combinatorial optimization problems which still resist any computational complexity characterization [6]. Despite decades of active research, no polynomial-time algorithm for it has yet been found. At the same time, while clearly belonging to $NP$, no proof has been provided that it is $NP$-complete. Indeed, there is strong evidence that this cannot be the case for, otherwise, the polynomial hierarchy would collapse [5]. The current belief is that the problem lies strictly between the $P$ and $NP$-complete classes.

Because of its theoretical as well as practical importance, the problem has attracted much attention in the neural network community, and various powerful heuristics have been developed [11, 18, 19, 20]. Following Hopfield and Tank's seminal work [10], the typical approach has been to write down a (continuous) energy function whose minimizers correspond to the (discrete) solutions being sought, and then construct a dynamical system which converges toward them. Almost invariably, all the algorithms developed so far are based on techniques borrowed from statistical mechanics, in particular mean field theory, which allow one to escape from poor

local solutions.

In this paper, we develop a new energy-minimization framework for the graph isomorphism problem which is based on the idea of reducing it to the maximum clique problem, another well-known combinatorial optimization problem. Central to our approach is a powerful result originally proved by Motzkin and Straus [13], and recently extended in various ways [3, 7, 16], which allows us to formulate the maximum clique problem in terms of an indefinite quadratic program. We then present a class of straightforward continuous- and discrete-time dynamical systems known in mathematical biology as *replicator equations*, and show how, thanks to their dynamical properties, they naturally suggest themselves as a useful heuristic for solving the proposed graph isomorphism program. The extensive experimental results presented show that, despite their simplicity and their inherent inability to escape from local optima, replicator dynamics are nevertheless competitive with more sophisticated deterministic annealing algorithms. The proposed formulation seems therefore a promising framework within which powerful continuous-based graph matching heuristics can be developed, and is in fact being employed for solving practical computer vision problems [17]. More details on the work presented here can be found in [15].

## 2   A QUADRATIC PROGRAM FOR GRAPH ISOMORPHISM

### 2.1   GRAPH ISOMORPHISM AS CLIQUE SEARCH

Let $G = (V, E)$ be an undirected graph, where $V$ is the set of vertices and $E \subseteq V \times V$ is the set of edges. The *order* of $G$ is the number of its vertices, and its *size* is the number of edges. Two vertices $i, j \in V$ are said to be *adjacent* if $(i, j) \in E$. The *adjacency matrix* of $G$ is the $n \times n$ symmetric matrix $A = (a_{ij})$ defined as follows: $a_{ij} = 1$ if $(i, j) \in E$, $a_{ij} = 0$ otherwise.

Given two graphs $G' = (V', E')$ and $G'' = (V'', E'')$ having the same order and size, an *isomorphism* between them is any bijection $\phi : V' \to V''$ such that $(i, j) \in E' \Leftrightarrow (\phi(i), \phi(j)) \in E''$, for all $i, j \in V'$. Two graphs are said to be *isomorphic* if there exists an isomorphism between them. The graph isomorphism problem is therefore to decide whether two graphs are isomorphic and, in the affirmative, to find an isomorphism. Barrow and Burstall [1] introduced the notion of an *association graph* as a useful auxiliary graph structure for solving general graph/subgraph isomorphism problems. The association graph derived from $G'$ and $G''$ is the undirected graph $G = (V, E)$, where $V = V' \times V''$ and

$$E = \{((i, h), (j, k)) \in V \times V \; : \; i \neq j, \; h \neq k, \; \text{and} \; (i, j) \in E' \Leftrightarrow (h, k) \in E''\} \; .$$

Given an arbitrary undirected graph $G = (V, E)$, a subset of vertices $C$ is called a *clique* if all its vertices are mutually adjacent, i.e., for all $i, j \in C$ we have $(i, j) \in E$. A clique is said to be *maximal* if it is not contained in any larger clique, and *maximum* if it is the largest clique in the graph. The *clique number*, denoted by $\omega(G)$, is defined as the cardinality of the maximum clique.

The following result establishes an equivalence between the graph isomorphism problem and the maximum clique problem (see [15] for proof).

**Theorem 2.1** *Let $G'$ and $G''$ be two graphs of order $n$, and let $G$ be the corresponding association graph. Then, $G'$ and $G''$ are isomorphic if and only if $\omega(G) = n$. In this case, any maximum clique of $G$ induces an isomorphism between $G'$ and $G''$, and vice versa.*

## 2.2   CONTINUOUS FORMULATION OF MAX-CLIQUE

Let $G = (V, E)$ be an arbitrary undirected graph of order $n$, and let $S_n$ denote the standard simplex of $\mathbb{R}^n$:

$$S_n = \left\{ \mathbf{x} \in \mathbb{R}^n \; : \; x_i \geq 0 \;\text{ for all } i = 1 \ldots n, \text{ and } \sum_{i=1}^{n} x_i = 1 \right\} .$$

Given a subset of vertices $C$ of $G$, we will denote by $\mathbf{x}^c$ its *characteristic vector* which is the point in $S_n$ defined as $x_i^c = 1/|C|$ if $i \in C$, $x_i^c = 0$ otherwise, where $|C|$ denotes the cardinality of $C$.

Now, consider the following quadratic function:

$$f(\mathbf{x}) \;\; = \;\; \mathbf{x}^T A \mathbf{x} \tag{1}$$

where "$T$" denotes transposition. The Motzkin-Straus theorem [13] establishes a remarkable connection between global (local) maximizers of $f$ in $S_n$ and maximum (maximal) cliques of $G$. Specifically, it states that a subset of vertices $C$ of a graph $G$ is a maximum clique if and only if its characteristic vector $\mathbf{x}^c$ is a global maximizer of the function $f$ in $S_n$. A similiar relationship holds between (strict) local maximizers and maximal cliques [7, 16].

One drawback associated with the original Motzkin-Straus formulation relates to the existence of spurious solutions, i.e., maximizers of $f$ which are not in the form of characteristic vectors [16]. In principle, spurious solutions represent a problem since, while providing information about the *order* of the maximum clique, do not allow us to extract the vertices comprising the clique. Fortunately, there is straightforward solution to this problem which has recently been introduced and studied by Bomze [3]. Consider the following regularized version of function $f$:

$$\hat{f}(\mathbf{x}) = \mathbf{x}^T A \mathbf{x} + \frac{1}{2} \mathbf{x}^T \mathbf{x} . \tag{2}$$

The following is the spurious-free counterpart of the original Motzkin-Straus theorem (see [3] for proof).

**Theorem 2.2** *Let $C$ be a subset of vertices of a graph $G$, and let $\mathbf{x}^c$ be its characteristic vector. Then the following statements hold:*

*(a) $C$ is a maximum clique of $G$ if and only if $\mathbf{x}^c$ is a global maximizer of $\hat{f}$ over the simplex $S_n$. Its order is then given by $|C| = 1/2(1 - f(\mathbf{x}^c))$.*

*(b) $C$ is a maximal clique of $G$ if and only if $\mathbf{x}^c$ is a local maximizer of $\hat{f}$ in $S_n$.*

*(c) All local (and hence global) maximizers of $\hat{f}$ over $S_n$ are strict.*

Unlike the Motzkin-Straus formulation, the previous result guarantees that *all* maximizers of $\hat{f}$ on $S_n$ are strict, and are characteristic vectors of maximal/maximum cliques in the graph. In an exact sense, therefore, a one-to-one correspondence exists between maximal cliques and local maximizers of $\hat{f}$ in $S_n$ on the one hand, and maximum cliques and global maximizers on the other hand.

## 2.3   A QUADRATIC PROGRAM FOR GRAPH ISOMORPHISM

Let $G'$ and $G''$ be two arbitrary graphs of order $n$, and let $A$ denote the adjacency matrix of the corresponding association graph, whose order is assumed to be $N$. The graph isomorphism problem is equivalent to the following program:

$$\begin{array}{ll} \text{maximize} & \hat{f}(\mathbf{x}) = \mathbf{x}^T (A + \frac{1}{2} I_N) \mathbf{x} \\ \text{subject to} & \mathbf{x} \in S_N \end{array} \tag{3}$$

More precisely, the following result holds, which is a straightforward consequence of Theorems 2.1 and 2.2.

**Theorem 2.3** *Let $G'$ and $G''$ be two graphs of order $n$, and let $\mathbf{x}^*$ be a global solution of program (3), where $A$ is the adjacency matrix of the association graph of $G'$ and $G''$. Then, $G'$ and $G''$ are isomorphic if and only if $\hat{f}(\mathbf{x}^*) = 1 - 1/2n$. In this case, any global solution to (3) induces an isomorphism between $G'$ and $G''$, and vice versa.*

In [15] we discuss the analogies between our objective function and those proposed in the literature (e.g., [18, 19]).

## 3 REPLICATOR EQUATIONS AND GRAPH ISOMORPHISM

Let $W$ be a non-negative $n \times n$ matrix, and consider the following dynamical system:

$$\frac{d}{dt}x_i(t) = x_i(t)\left(\pi_i(t) - \sum_{j=1}^{n} x_j(t)\pi_j(t)\right) , \quad i = 1 \ldots n \tag{4}$$

where $\pi_i(t) = \sum_{j=1}^{n} w_{ij}x_j(t)$, $i = 1 \ldots n$, and its discrete-time counterpart:

$$x_i(t+1) = \frac{x_i(t)\pi_i(t)}{\sum_{j=1}^{n} x_j(t)\pi_j(t)} , \quad i = 1 \ldots n . \tag{5}$$

It is readily seen that the simplex $S_n$ is invariant under these dynamics, which means that every trajectory starting in $S_n$ will remain in $S_n$ for all future times.

Both (4) and (5) are called *replicator equations* in theoretical biology, since they are used to model evolution over time of relative frequencies of interacting, self-replicating entities [9]. The discrete-time dynamical equations turn also out to be a special case of a general class of dynamical systems introduced by Baum and Eagon [2] in the context of Markov chain theory.

**Theorem 3.1** *If $W$ is symmetric, then the quadratic polynomial $F(\mathbf{x}) = \mathbf{x}^T W \mathbf{x}$ is strictly increasing along any non-constant trajectory of both continuous-time (4) and discrete-time (5) replicator equations. Furthermore, any such trajectory converges to a (unique) stationary point. Finally, a vector $\mathbf{x} \in S_n$ is asymptotically stable under (4) and (5) if and only if $\mathbf{x}$ is a strict local maximizer of $F$ on $S_n$.*

The previous result is known in mathematical biology as the Fundamental Theorem of Natural Selection [9, 21]. As far as the discrete-time model is concerned, it can be regarded as a straightforward implication of the more general Baum-Eagon theorem [2]. The fact that all trajectories of the replicator dynamics converge to a stationary point is proven in [12].

Recently, there has been much interest in evolutionary game theory around the following exponential version of replicator equations, which arises as a model of evolution guided by imitation [8, 21]:

$$\frac{d}{dt}x_i(t) = x_i(t)\left(\frac{e^{\kappa\pi_i(t)}}{\sum_{j=1}^{n} x_j(t)e^{\kappa\pi_j(t)}} - 1\right) , \quad i = 1 \ldots n \tag{6}$$

where $\kappa$ is a positive constant. As $\kappa$ tends to 0, the orbits of this dynamics approach those of the standard, first-order replicator model (4), slowed down by the factor

$\kappa$. Hofbauer [8] has recently proven that when the matrix $W$ is symmetric, the quadratic polynomial $F$ defined in Theorem 3.1 is also strictly increasing, as in the first-order case. After discussing various properties of this, and more general dynamics, he concluded that the model behaves essentially in the same way as the standard replicator equations, the only difference being the size of the basins of attraction around stable equilibria. A customary way of discretizating equation (6) is given by the following difference equations:

$$x_i(t+1) = \frac{x_i(t)e^{\kappa\pi_i(t)}}{\sum_{j=1}^{n} x_j(t)e^{\kappa\pi_j(t)}} \ , \quad i = 1\ldots n \tag{7}$$

which enjoys many of the properties of the first-order system (5), e.g., they have the same set of equilibria.

The properties discussed above naturally suggest using replicator equations as a useful heuristic for the graph isomorphism problem. Let $G'$ and $G''$ be two graphs of order $n$, and let $A$ denote the adjacency matrix of the corresponding $N$-vertex association graph $G$. By letting

$$W = A + \frac{1}{2}I_N$$

we know that the replicator dynamical systems, starting from an arbitrary initial state, will iteratively maximize the function $\hat{f}(\mathbf{x}) = \mathbf{x}^T(A + \frac{1}{2}I_N)\mathbf{x}$ in $S_N$, and will eventually converge to a strict local maximizer which, by virtue of Theorem 2.2 will then correspond to the characteristic vector of a maximal clique in the association graph. This will in turn induce an isomorphism between two subgraphs of $G'$ and $G''$ which is "maximal," in the sense that there is no other isomorphism between subgraphs of $G'$ and $G''$ which includes the one found. Clearly, in theory there is no guarantee that the converged solution will be a *global* maximizer of $\hat{f}$, and therefore that it will induce an isomorphism between the two original graphs. Previous work done on the maximum clique problem [4, 14], and also the results presented in this paper, however, suggest that the basins of attraction of global maximizers are quite large, and very frequently the algorithm converges to one of them.

## 4  EXPERIMENTAL RESULTS

In the experiments reported here, the discrete-time replicator equation (5) and its exponential counterpart (7) with $\kappa = 10$ were used. The algorithms were started from the barycenter of the simplex and they were stopped when either a maximal clique was found or the distance between two successive points was smaller than a fixed threshold, which was set to $10^{-17}$. In the latter case the converged vector was randomly perturbed, and the algorithm restarted from the perturbed point. Because of the one-to-one correspondence between local maximizers and maximal cliques, this situation corresponds to convergence to a saddle point. All the experiments were run on a Sparc20.

Undirected 100-vertex random graphs were generated with expected connectivities ranging from 1% to 99%. For each connectivity value, 100 graphs were produced and each of them had its vertices randomly permuted so as to obtain a pair of isomorphic graphs. Overall, therefore, 1500 pairs of isomorphic graphs were used. Each pair was given as input to the replicator models and, after convergence, a success was recorded when the cardinality of the returned clique was equal to the order of the graphs given as input (i.e., 100).[1] Because of the stopping criterion employed, this

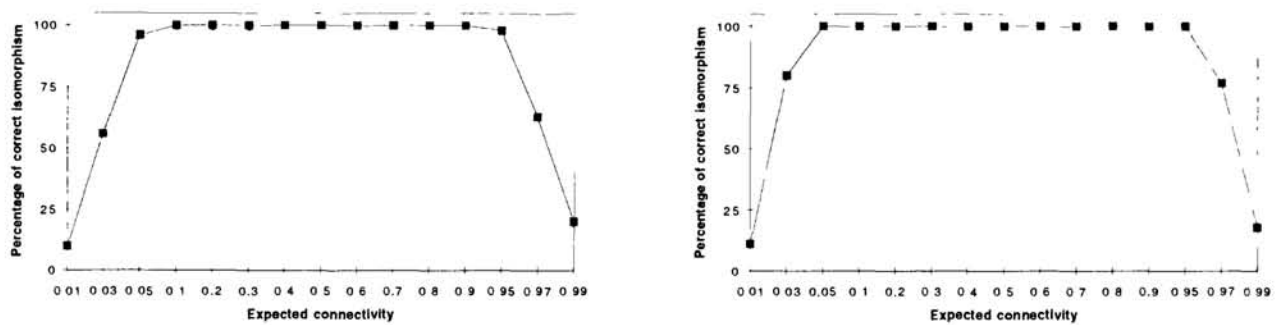

Figure 1: Percentage of correct isomorphisms obtained using the first-order (left) and the exponential (right) replicator equations, as a function of the expected connectivity.

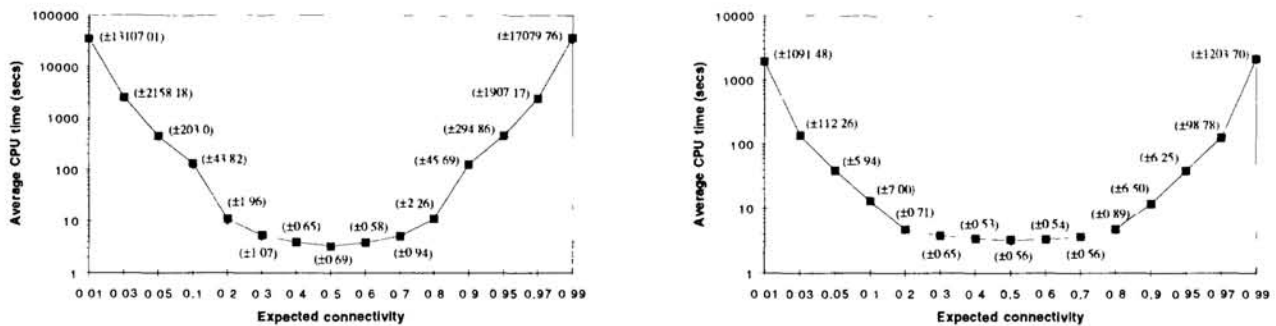

Figure 2: Average computational time taken by the first-order (left) and the exponential (right) replicator equations, as a function of the expected connectivity. The vertical axes are in logarithmic scale, and the numbers in parentheses represent the standard deviation.

guarantees that a maximum clique, and therefore a correct isomorphism, was found. The proportion of successes as a function of the expected connectivities for both replicator models is plotted in Fig. 1, whereas Fig. 2 shows the average CPU time taken by the two algorithms to converge (in logarithmic scale). Notice how the exponential replicator system (7) is dramatically faster and also performs better than the first-order model (5).

These results are significantly superior to those reported by Simić [20] who obtained poor results at connectivities less than 40% even on smaller graphs (i.e., up to 75 vertices). They also compare favorably with the results obtained more recently by Rangarajan *et al.* [18] on 100-vertex random graphs for connectivities up to 50%. Specifically, at 1% and 3% connectivities they report a percentage of correct isomorphisms of about 30% and 0%, respectively. Using our approach we obtained, on the same kind of graphs, a percentage of success of 80% and 11%, respectively. Rangarajan and Mjolsness [19] also ran experiments on 100-vertex random graphs with various connectivities, using a powerful Lagrangian relaxation network. Except for a few instances, they always obtained a correct solution. The computational time required by their model, however, turns out to largely exceed ours. As an example, the average time taken by their algorithm to match two 100-vertex 50%-connectivity graphs was about 30 minutes on an SGI workstation. As shown in Fig. 2, we obtained identical results in about 3 seconds.

It should be emphasized that all the algorithms mentioned above do incorporate sophisticated annealing mechanisms to escape from poor local minima. By contrast, in the presented work no attempt was made to prevent the algorithms from converging to such solutions.

**Acknowledgments.** This work has been done while the author was visiting the Department of Computer Science at the Yale University. Funding for this research has been provided by the Consiglio Nazionale delle Ricerche, Italy. The author would like to thank I. M. Bomze, A. Rangarajan, K. Siddiqi, and S. W. Zucker for many stimulating discussions.

## Footnotes

[1]Due to the high computational time required, in the 1% and 99% cases the first-order replicator algorithm (5) was tested only on 10 pairs, instead of 100.

# References

[1] H. G. Barrow and R. M. Burstall, "Subgraph isomorphism, matching relational structures and maximal cliques," *Inform. Process. Lett.*, vol. 4, no. 4, pp. 83–84, 1976.

[2] L. E. Baum and J. A. Eagon, "An inequality with applications to statistical estimation for probabilistic functions of Markov processes and to a model for ecology," *Bull. Amer. Math. Soc.*, vol. 73, pp. 360–363, 1967.

[3] I. M. Bomze, "Evolution towards the maximum clique," *J. Global Optim.*, vol. 10, pp. 143–164, 1997.

[4] I. M. Bomze, M. Pelillo, and R. Giacomini, "Evolutionary approach to the maximum clique problem: Empirical evidence on a larger scale," in *Developments in Global Optimization*, I. M. Bomze *et al.*, eds., Kluwer, The Netherlands, 1997, pp. 95–108.

[5] R. B. Boppana, J. Hastad, and S. Zachos, "Does co-NP have short interactive proofs?" *Inform. Process. Lett.*, vol. 25, pp. 127–132, 1987.

[6] M. R. Garey and D. S. Johnson, *Computers and Intractability: A Guide to the Theory of NP-Completeness.* Freeman, San Francisco, CA, 1979.

[7] L. E. Gibbons, D. W. Hearn, P. M. Pardalos, and M. V. Ramana, "Continuous characterizations of the maximum clique problem," *Math. Oper. Res.*, vol. 22, no. 3, pp. 754–768, 1997.

[8] J. Hofbauer, "Imitation dynamics for games," Collegium Budapest, preprint, 1995.

[9] J. Hofbauer and K. Sigmund, *The Theory of Evolution and Dynamical Systems.* Cambridge University Press, Cambridge, UK, 1988.

[10] J. J. Hopfield and D. W. Tank, "Neural computation of decisions in optimization problems," *Biol. Cybern.*, vol. 52, pp. 141–152, 1985.

[11] R. Kree and A. Zippelius, "Recognition of topological features of graphs and images in neural networks," *J. Phys. A: Math. Gen.*, vol. 21, pp. L813–L818, 1988.

[12] V. Losert and E. Akin, "Dynamics of games and genes: Discrete versus continuous time," *J. Math. Biol.*, vol. 17, pp. 241–251, 1983.

[13] T. S. Motzkin and E. G. Straus, "Maxima for graphs and a new proof of a theorem of Turán," *Canad. J. Math.*, vol. 17, pp. 533–540, 1965.

[14] M. Pelillo, "Relaxation labeling networks for the maximum clique problem," *J. Artif. Neural Networks*, vol. 2, no. 4, pp. 313–328, 1995.

[15] M. Pelillo, "Replicator equations, maximal cliques, and graph isomorphism," *Neural Computation*, to appear.

[16] M. Pelillo and A. Jagota, "Feasible and infeasible maxima in a quadratic program for maximum clique," *J. Artif. Neural Networks*, vol. 2, no. 4, pp. 411–420, 1995.

[17] M. Pelillo, K. Siddiqi, and S. W Zucker, "Matching hierarchical structures using association graphs," in *Computer Vision—ECCV'98, Vol. II*, H. Burkhardt and B. Neumann, eds., Springer-Verlag, Berlin, 1998, pp. 3–16.

[18] A. Rangarajan, S. Gold, and E. Mjolsness, "A novel optimizing network architecture with applications," *Neural Computation*, vol. 8, pp. 1041–1060, 1996.

[19] A. Rangarajan and E. Mjolsness, "A Lagrangian relaxation network for graph matching," *IEEE Trans. Neural Networks*, vol. 7, no. 6, pp. 1365–1381, 1996.

[20] P. D. Simić, "Constrained nets for graph matching and other quadratic assignment problems," *Neural Computation*, vol. 3, pp. 268–281, 1991.

[21] J. W. Weibull, *Evolutionary Game Theory.* MIT Press, Cambridge, MA, 1995.